# Decomposing Isotonic Regression for Efficiently Solving Large Problems

**Ronny Luss**
Dept. of Statistics and OR
Tel Aviv University
ronnyluss@gmail.com

**Saharon Rosset**
Dept. of Statistics and OR
Tel Aviv University
saharon@post.tau.ac.il

**Moni Shahar**
Dept. of Electrical Eng.
Tel Aviv University
moni@eng.tau.ac.il

## Abstract

A new algorithm for isotonic regression is presented based on recursively partitioning the solution space. We develop efficient methods for each partitioning subproblem through an equivalent representation as a network flow problem, and prove that this sequence of partitions converges to the global solution. These network flow problems can further be decomposed in order to solve very large problems. Success of isotonic regression in prediction and our algorithm's favorable computational properties are demonstrated through simulated examples as large as $2 \times 10^5$ variables and $10^7$ constraints.

## 1 Introduction

Assume we have a set of $n$ data observations $(x_1, y_1), ..., (x_n, y_n)$, where $x \in \mathcal{X}$ (usually $\mathcal{X} = \mathbb{R}^p$) is a vector of covariates or independent variables, $y \in \mathbb{R}$ is the response, and we wish to fit a model $\hat{f} : \mathcal{X} \to \mathbb{R}$ to describe the dependence of $y$ on $x$, i.e., $y \approx \hat{f}(x)$. Isotonic regression is a non-parametric modeling approach which only restricts the fitted model to being monotone in all independent variables [1]. Define $\mathcal{G}$ as the family of isotonic functions, that is, $g \in \mathcal{G}$ satisfies

$$x_1 \preceq x_2 \Rightarrow g(x_1) \leq g(x_2),$$

where the partial order $\preceq$ here will usually be the standard Euclidean one, i.e., $x_1 \preceq x_2$ if $x_{1j} \leq x_{2j}$ $\forall j$. Given these definitions, isotonic regression solves

$$\hat{f} = \arg\min_{g \in \mathcal{G}} \|\mathbf{y} - g(\mathbf{x})\|^2. \tag{1}$$

As many authors have noted, the optimal solution to this problem comprises a partitioning of the space $\mathcal{X}$ into regions obeying a monotonicity property with a constant fitted to $\hat{f}$ in each region.

It is clear that isotonic regression is a very attractive model for situations where monotonicity is a reasonable assumption, but other common assumptions like linearity or additivity are not. Indeed, this formulation has found useful applications in biology [2], medicine [3], statistics [1] and psychology [4], among others. Practicality of isotonic regression has already been demonstrated in various fields and in this paper we focus on algorithms for computing isotonic regressions on large problems.

An equivalent formulation of $L_2$ isotonic regression seeks an optimal isotonic fit $\hat{y}_i$ at every point by solving

$$
\begin{aligned}
\text{minimize} \quad & \sum_{i=1}^{n} (\hat{y}_i - y_i)^2 \\
\text{subject to} \quad & \hat{y}_i \leq \hat{y}_j \qquad \forall (i, j) \in \mathcal{I}
\end{aligned}
\tag{2}
$$

where $\mathcal{I}$ denotes a set of isotonic constraints. This paper assumes that $\mathcal{I}$ contains no redundant constraints, i.e. $(i, j), (j, k) \in \mathcal{I} \Rightarrow (i, k) \notin \mathcal{I}$. Problem (2) is a quadratic program subject to

simple linear constraints, and, according to a literature review, appears to be largely ignored due to computational difficulty on large problems. The worst case $O(n^4)$ complexity (a large overstatement in practice as will be shown) has resulted in overlooking the results that follow [5, 6].

The discussion of isotonic regression originally focused on the case $x \in \mathbb{R}$, where $\preceq$ denoted a complete order [4]. For this case, the well known pooled adjacent violators algorithm (PAVA) efficiently solves the isotonic regression problem. For the partially ordered case, many different algorithms have been developed over the years, with most early efforts concentrated on generalizations of PAVA [7, 5]. These algorithms typically have no polynomial complexity guarantees and are impractical when data size exceed a few thousand observations. Problem (1) can also be treated as a separable quadratic program subject to simple linear equality constraints. Such was done, for example, in [8], which applies active set methods to solve the problem. While such algorithms can often be efficient in practice, the algorithm of [8] gives no complexity guarantees. Related algorithms in [9] to those described here were applied to problems for scheduling reorder intervals in production systems and are of complexity $O(n^4)$ and connections to isotonic regression can be made through [1]. Interior point methods are another tool for solving Problem (1), and have time complexity guarantees of $O(n^3)$ when the number of constraints is on the same order as the number of variables (see [10]). However, the excessive memory requirements of interior point methods from solving large systems of linear equations typically make them impractical for large data sizes. Recently, [6] and [11] gave an $O(n^2)$ *approximate* generalized PAVA algorithm, however solution quality can only be demonstrated via experimentation. An even better complexity of $O(n \log n)$ can be obtained for the optimal solution when the isotonic constraints take a special structure such as a tree, e.g. [12].

## 1.1 Contribution

Our novel approach to isotonic regression offers an exact solution of (1) with a complexity bounded by $O(n^4)$, but acts on the order of $O(n^3)$ for practical problems. We demonstrate here that it accommodates problems with tens of thousands of observations, or even more with our decomposition. The main goal of this paper is to make isotonic regression a reasonable computational tool for large data sets, as the assumptions in this framework are very applicable in real-world applications. Our framework solves quadratic programs with $2 \times 10^5$ variables and more than $10^7$ constraints, a problem of size not solved anywhere in previous isotonic regression literature, and with the decomposition detailed below, even larger problems can be solved.

The paper is organized as follows. Section 2 describes a partitioning algorithm for isotonic regression and proves convergence to the globally optimal solution. Section 3 explains how the subproblems (creating a single partition) can be solved efficiently and decomposed in order to solve large-scale problems. Section 4 demonstrates that the partitioning algorithm is significantly better in practice than the $O(n^4)$ worst-case complexity. Finally, Section 5 gives numerical results and demonstrates favorable predictive performance on large simulated data sets and Section 6 concludes with future directions.

### Notation
The *weight* of a set of points $A$ is defined as $\overline{y}_A = \frac{1}{|A|} \sum_{i \in A} y_i$. A subset $\mathcal{U}$ of $A$ is an *upper set* of $A$ if $x \in \mathcal{U}, y \in A, x \prec y \Rightarrow y \in \mathcal{U}$. A set $B \subseteq A$ is defined as a block of $A$ if $\overline{y}_{\mathcal{U} \cap B} \leq \overline{y}_B$ for each upper set $\mathcal{U}$ of $A$ such that $\mathcal{U} \cap B \neq \{\}$. A general block $A$ is considered a block of the entire space. For two blocks $A$ and $B$, we denote $A \preceq B$ if $\exists x \in A, y \in B$ such that $x \preceq y$ and $\nexists x \in A, y \in B$ such that $y \preceq x$ (i.e. there is at least one comparable pair of points that satisfy the direction of isotonicity). $A$ and $B$ are then said to be isotonic blocks (or obey isotonicity). A group of nodes $X$ *majorizes* (*minorizes*) another group $Y$ if $X \succeq Y$ ($X \preceq Y$). A group $X$ is a *majorant* (*minorant*) of $X \cup A$ where $A = \cup_{i=1}^{k} A_i$ if $X \npreceq A_i$ ($X \nsucceq A_i$) $\forall i = 1 \ldots k$.

## 2 Partitioning Algorithm

We first describe the structure of the classic $L_2$ isotonic regression problem and continue to detail the partitioning algorithm. The section concludes by proving convergence of the algorithm to the globally optimal isotonic regression solution.

## 2.1 Structure

Problem (2) is a quadratic program subject to simple linear constraints. The structure of the optimal solution to (2) is well-known. Observations are divided into $k$ groups where the fits in each group take the group mean observation value. This can be seen through the equations given by the following Karush-Kuhn-Tucker (KKT) conditions:

(a) $\hat{y}_i = y_i - \dfrac{1}{2}\left( \sum\limits_{j:(i,j)\in\mathcal{I}} \lambda_{ij} - \sum\limits_{j:(j,i)\in\mathcal{I}} \lambda_{ji} \right)$

(b) $\hat{y}_i \leq \hat{y}_j \; \forall (i,j) \in \mathcal{I}$

(c) $\lambda_{ij} \geq 0 \; \forall (i,j) \in \mathcal{I}$

(d) $\lambda_{ij}(\hat{y}_i - \hat{y}_j) = 0 \; \forall (i,j) \in \mathcal{I}.$

This set of conditions exposes the nature of the optimal solution, since condition (d) implies that $\lambda_{ij} > 0 \Rightarrow \hat{y}_i = \hat{y}_j$. Hence $\lambda_{ij}$ can be non-zero only within blocks in the isotonic solution which have the same fitted value. For observations in different blocks, $\lambda_{ij} = 0$. Furthermore, the fit within each block is trivially seen to be the average of the observations in the block, i.e. the fits minimize the block's squared loss. Thus, we get the familiar characterization of the isotonic regression problem as one of finding a division into isotonic blocks.

## 2.2 Partitioning

In order to take advantage of the optimal solution's structure, we propose solving the isotonic regression problem (2) as a sequence of subproblems that divides a group of nodes into two groups at each iteration. An important property of our partitioning approach is that nodes separated at one iteration are never rejoined into the same group in future iterations. This gives a clear bound on the total number of iterations in the worst case.

We now describe the partitioning criterion used for each subproblem. Suppose a current block $\mathcal{V}$ is optimal and thus $\hat{y}_i^* = \overline{y}_{\mathcal{V}} \; \forall i \in \mathcal{V}$. From condition (a) of the KKT conditions, we define the net outflow of a group $\mathcal{V}$ as $\sum_{i \in \mathcal{V}} (y_i - \hat{y}_i)$. Finding two groups within $\mathcal{V}$ such that the net outflow from the higher group is greater than the net outflow from the lower group should be infeasible, according to the KKT conditions. The partition here looks for two such groups. Denote by $\mathcal{C}_{\mathcal{V}}$ the set of all feasible (i.e. isotonic) cuts through the network defined by nodes in $\mathcal{V}$. A cut is called isotonic if the two blocks created by the cut are isotonic. The optimal cut is determined as the cut that solves the problem

$$\max_{c \in \mathcal{C}_{\mathcal{V}}} \sum_{i \in \mathcal{V}_c^+} (y_i - \overline{y}_{\mathcal{V}}) - \sum_{i \in \mathcal{V}_c^-} (y_i - \overline{y}_{\mathcal{V}}) \tag{3}$$

where $\mathcal{V}_c^-(\mathcal{V}_c^+)$ is the group on the lower (upper) side of the edges of cut $c$. In terms of isotonic regression, the optimal cut is such that the difference in the sum of the normalized fits $(y_i - \overline{y}_{\mathcal{V}})$ at each node of a group is maximized. If this maximized difference is zero, then the group must be an optimal block. The optimal cut problem (3) can also be written as the binary program

$$\begin{aligned}
\text{maximize} \quad & \sum_i x_i (y_i - \overline{y}_{\mathcal{V}}) \\
\text{subject to} \quad & x_i \leq x_j && \forall (i,j) \in \mathcal{I} \\
& x_i \in \{-1, +1\} && \forall i \in \mathcal{V}.
\end{aligned} \tag{4}$$

Well-known results from [13] (due to the fact that the constraint matrix is totally unimodular) say that the following relaxation to this binary program is optimal with $x^*$ on the boundary, and hence the optimal cut can be determined by solving the linear program

$$\begin{aligned}
\text{maximize} \quad & z^T x \\
\text{subject to} \quad & x_i \leq x_j && \forall (i,j) \in \mathcal{I} \\
& -1 \leq x_i \leq 1 && \forall i \in \mathcal{V}
\end{aligned} \tag{5}$$

where $z_i = y_i - \overline{y}_{\mathcal{V}}$. This group-wise partitioning operation is the basis for our partitioning algorithm which is explicitly given in Algorithm 1. It starts with all observations as one group (i.e., $\mathcal{V} = \{1, \ldots, n\}$), and recursively splits each group optimally by solving subproblem (5). At each

iteration, a list $\mathcal{C}$ of potential optimal cuts for each group generated thus far is maintained, and the cut among them with the highest objective value is performed. The list $\mathcal{C}$ is updated with the optimal cuts in both sub-groups generated. Partitioning ends whenever the solution to (5) is trivial (i.e., no split is found because the group is a block). As proven next, this algorithm terminates with the optimal global (isotonic) solution to the isotonic regression problem (2).

---

**Algorithm 1** Paritioning Algorithm

---

**Require:** Observations $y_1, \ldots, y_n$ and partial order $\mathcal{I}$.
**Require:** $\mathcal{V} = \{\{1, \ldots, n\}\}, \mathcal{C} = \{(0, \{1, \ldots, n\}, \{\})\}, \mathcal{W} = \{\}$.
1: **while** $\mathcal{V} \neq \{\}$ **do**
2:     Let $(val, w^-, w^+) \in \mathcal{C}$ be the potential cut with largest $val$.
3:     Update $\mathcal{V} = (\mathcal{V} \setminus (w^- \cup w^+)) \cup \{w^-, w^+\}, \mathcal{C} = \mathcal{C} \setminus (val, w^-, w^+)$ .
4:     **for all** $v \in \{w^-, w^+\}$ **do**
5:        Set $z_i = y_i - \overline{y}_v \ \forall i \in v$ where $\overline{y}_v$ is the mean of observations in $v$.
6:        Solve LP (5) with input $z$ and get $x^*$.
7:        **if** $x_1^* = \ldots = x_n^*$ (group is optimally divided) **then**
8:           Update $\mathcal{V} = \mathcal{V} \setminus v$ and $\mathcal{W} = \mathcal{W} \cup v$.
9:        **else**
10:          Let $v^- = \{i : x_i^* = -1\}, v^+ = \{i : x_i^* = +1\}$.
11:          Update $\mathcal{C} = \mathcal{C} \cup \{(z^T x^*, v^-, v^+)\}$
12:        **end if**
13:     **end for**
14: **end while**
15: **return** $\mathcal{W}$ the optimal groups

---

## 2.3 Convergence

Theorem 1 next states the main result that allows for a no-regret partitioning algorithm for isotonic regression. This will lead to our convergence result. We assume that group $\mathcal{V}$ is isotonic (i.e. has no holes) and is the union of optimal blocks.

**Theorem 1** *Assume a group $\mathcal{V}$ is a union of blocks from the optimal solution to problem (2). Then a cut made by solving (5) does not cut through any block in the global optimal solution.*

**Proof.** The following is a brief sketch of the proof idea: Let $\mathcal{M}$ be the union of $K$ optimal blocks in $\mathcal{V}$ that get broken by the cut. Define $M_1$ ($M_K$) to be a minorant (majorant) block in $\mathcal{M}$. For each $M_k$ define $M_k^L$ ($M_k^U$) as the groups in $M_k$ below (above) the algorithm cut. Using the definitions of how the algorithm makes partitions, the following two consequences can be proven: (1) $\overline{y}_{M_1} < \overline{y}_{M_K}$ by optimality (i.e. according to KKT conditions) and isotonicity and (2) $\overline{y}_{M_1} > \overline{y}_{\mathcal{V}}$ and $\overline{y}_{M_K} < \overline{y}_{\mathcal{V}}$. This is proven by showing that $\overline{y}_{M_1^U} > \overline{y}_{\mathcal{V}}$, because otherwise the $M_1^U$ block would be on the lower side of the cut, resulting in $M_1$ being on the lower side of the cut, and thus $\overline{y}_{M_1} > \overline{y}_{\mathcal{V}}$ since $\overline{y}_{M_1^L} > \overline{y}_{M_1^U}$ by the optimality assumption on block $M_1$ (with symmetric arguments for $M_K$). This leads to the contradiction $\overline{y}_{\mathcal{V}} < \overline{y}_{M_1} < \overline{y}_{M_K} < \overline{y}_{\mathcal{V}}$, and hence $\mathcal{M}$ must be empty. ■

Since Algorithm 1 starts with $\mathcal{V} = \{1, ..., n\}$ which is a union of (all) optimal blocks, we can conclude from this theorem that partitions never cut an optimal block. The following corollary is then a direct consequence of repeatedly applying Theorem 1 in Algorithm 1:

**Corollary 2** *Algorithm 1 converges to the global optimal solution of (2) with no regret (i.e. without having to rejoin observations that are divided at a previous iteration).*

## 3 Efficient solutions of the subproblems

Linear program (5) has a special structure that can be taken advantage of in order to solve larger problems faster. We first show why these problems can be solved faster than typical linear programs, followed by a novel decomposition of the structure that allows problems of extremely large size to be solved efficiently.

## 3.1 Network flow problems

The dual to Problem (2) is a network flow problem with quadratic objective. The network flow constraints are identical to those in (6) below, but the objective is $\frac{1}{4}\sum_{i=1}^{n}(s_i^2 + t_i^2)$, which, to the author's knowledge, currently still precludes this dual from being efficiently solved with special network algorithms.

While this structure does not help solve directly the quadratic program, the network structure allows the linear program for the subproblems to be solved very efficiently. The dual program to (5) is

$$
\begin{aligned}
\text{minimize} \quad & \sum_{i \in \mathcal{V}} (s_i + t_i) \\
\text{subject to} \quad & \sum_{j:(i,j) \in \mathcal{I}} \lambda_{ij} - \sum_{j:(j,i) \in \mathcal{I}} \lambda_{ji} - s_i + t_i = z_i \quad \forall i \in \mathcal{V} \\
& \lambda, s, t \geq 0
\end{aligned}
\tag{6}
$$

where again $z_i = y_i - \overline{y}_{\mathcal{V}}$. Linear program (6) is a network flow problem with $|\mathcal{V}| + 2$ nodes and $|\mathcal{I}| + 2|\mathcal{V}|$ arcs. Variable $s$ denotes links directed from a source node into each other node, while $t$ denotes links connecting each node into a sink node. The network flow problem here minimizes the total sum of flow over links from the source and into the sink with the goal to leave $z_i$ units of flow at each node $i \in \mathcal{V}$. Note that this is very similar to the network flow problem solved in [14] where $z_i$ there represents the classification performance on node $i$. Specialized simplex methods for such network flow problems are typically much faster ([15] documents an average speedup factor of 10 to 100 over standard simplex solvers) due to several reasons such as simpler operations on network data structures rather than maintaining and operating on the simplex tableau (see [16] for an overview of network simplex methods).

## 3.2 Large-scale decompositions

In addition to having a very efficient method for solving this network flow problem, further enhancements can be made on extremely large problems of similar structure that might suffer from memory problems. It is already assumed that no redundant arcs exist in $\mathcal{I}$ (i.e. $(i,j), (j,k) \in \mathcal{I} \Rightarrow (i,k) \notin \mathcal{I}$). One simple reduction involves eliminating negative (positive) nodes, i.e. nodes with $z_i < 0$ ($z_i \geq 0$) where where $z_i = y_i - \overline{y}_{\mathcal{V}}$, that are bounded only from above (below). It is trivial to observe that these nodes will be be equal to $-1$ ($+1$) in the optimal solution and that eliminating them does not affect solving (5) without them. However, in practice, this trivial reduction has a computationally minimal affect on large data sets. These reductions were also discussed in [14].

We next consider a novel reduction for the primal linear program (5). The main idea is that it can be solved through a sequence of smaller linear programs that reduce the total size of the full linear program on each iteration. Consider a minorant group of nodes $\mathcal{J} \subseteq \mathcal{V}$ and the subset of arcs $\mathcal{I}_{\mathcal{J}} \subseteq \mathcal{I}$ connecting them. Solving problem (5) on this reduced network with the original input $z$ divides the nodes in $\mathcal{J}$ into a lower and upper group, denoted $\mathcal{J}_L$ and $\mathcal{J}_U$. Nodes in $\mathcal{J}_L$ are not bounded from above and will be in the lower group of the full problem solved on $\mathcal{V}$. In addition, the same problem solved on the remaining nodes in $\mathcal{V} \setminus \mathcal{J}_L$ will give the optimal solutions to these nodes. This is formalized in Proposition 3.

**Proposition 3** *Let $\mathcal{J} \subseteq \mathcal{V}$ be a minorant group of nodes in $\mathcal{V}$. Let $w^*$ and $x^*$ be optimal solutions to Problem (5) on the reduced set $\mathcal{J}$ and full set $\mathcal{V}$ of nodes, respectively. If $w_i^* = -1$, then $x_i^* = -1$ $\forall i \in \mathcal{J}$. The optimal solution for the remaining nodes $(\mathcal{V} \setminus \mathcal{J})$ can be found by solving (5) over only those nodes. The same claims can be made when $\mathcal{J} \subseteq \mathcal{V}$ is a majorant group of nodes in $\mathcal{V}$ where instead $w_i^* = +1 \Rightarrow x_i^* = +1 \ \forall i \in \mathcal{J}$.*

**Proof.** Denote $\mathcal{W}$ the set of nodes such that $w_i^* = -1$ and $\hat{\mathcal{W}} = \mathcal{V} \setminus \mathcal{W}$. Clearly, the solution to Problem (5) over nodes in $\mathcal{W}$ has the solution with all variables equal to $-1$. Problem (5) can be written in the following form with separable objective:

$$
\begin{aligned}
\text{maximize} \quad & \sum_{i \in \mathcal{W}} z_i x_i + \sum_{i \in \mathcal{V} \setminus \mathcal{W}} z_i x_i \\
\text{subject to} \quad & x_i \leq x_j && \forall (i,j) \in \mathcal{I}, i, j \in \mathcal{W} \\
& x_i \leq x_j && \forall (i,j) \in \mathcal{I}, i \in \mathcal{V}, j \in \mathcal{V} \setminus \mathcal{W} \\
& -1 \leq x_i \leq 1 && \forall i \in \mathcal{V}
\end{aligned}
\tag{7}
$$

Start with an initial solution $x_i = 1 \; \forall i \in \mathcal{V}$. Variables in $\mathcal{W}$ can be optimized over first and by assumption have the optimal value with all variables equal to $-1$. Optimization over variables in $\hat{\mathcal{W}}$ is not bounded from below by variables in $\mathcal{W}$ since those variables are all at the lower bound. Hence the optimal solution to variables in $\hat{\mathcal{W}}$ is given by optimizing over only these variables. The result for minorant groups follows. The final claim is easily argued in the same way as for the minorant groups. ∎

Given Proposition 3, Algorithm 2, which iteratively solves (5), can be stated. The subtrees are built as follows. First, an upper triangular adjacency matrix $C$ can be constructed to represent $\mathcal{I}$, where $C_{ij} = 1$ if $x_i \le x_j$ is an isotonic constraint and $C_{ij} = 0$ otherwise. A minorant (majorant) subtree with $k$ nodes is then constructed as the upper left (lower right) $k \times k$ sub-matrix of C.

---

**Algorithm 2** Iterative algorithm for linear program (5)

---

**Require:** Observations $y_1, \ldots, y_n$ and partial order $\mathcal{I}$.
**Require:** $MAXSIZE$ of problem to be solved by general LP solver
**Require:** $\mathcal{V} = \{1, \ldots, n\}, \mathcal{L} = \mathcal{U} = \{\}$.
 1: **while** $|\mathcal{V}| \ge MAXSIZE$ **do**
 2:     ELIMINATE A MINORANT SET OF NODES:
 3:     Build a minorant subtree $\mathcal{T}$.
 4:     Solve linear program (5) on $\mathcal{T}$ and get solution $\hat{y} \in \{-1, +1\}^{|\mathcal{T}|}$.
 5:     $\mathcal{L} = \mathcal{L} \cup \{v \in \mathcal{T} : \hat{y}_v = -1\}, \mathcal{V} = \mathcal{V} \setminus \{v \in \mathcal{T} : \hat{y}_v = -1\}$.
 6:     ELIMINATE A MAJORANT SET OF NODES:
 7:     Build majorant subtree $\mathcal{T}$.
 8:     Solve linear program (5) on $\mathcal{T}$ and get solution $\hat{y} \in \{-1, +1\}^{|\mathcal{T}|}$.
 9:     $\mathcal{U} = \mathcal{U} \cup \{v \in \mathcal{T} : \hat{y}_v = +1\}, \mathcal{V} = \mathcal{V} \setminus \{v \in \mathcal{T} : \hat{y}_v = +1\}$.
10: **end while**
11: Solve linear program (5) on $\mathcal{V}$ and get solution $\hat{y} \in \{-1, +1\}^{|\mathcal{V}|}$.
12: $\mathcal{L} = \mathcal{L} \cup \{v \in \mathcal{T} : \hat{y}_v = -1\}, \mathcal{U} = \mathcal{U} \cup \{v \in \mathcal{T} : \hat{y}_v = +1\}$.

---

The computational bottleneck of Algorithm 2 is solving linear program (5), which is done efficiently by solving the dual network flow problem (6). This shows that, if the first network flow problem is too large to solve, it can be solved by a sequence of smaller network flow problems as illustrated in Figure 1. Lemma 4 below proves that this reduction optimally solves the full problem (5). In the worst case, many network flow problems will be solved until the original full-size network flow problem is solved. However, in practice on large problems, this artifact is never observed. Computational performance of this reduction is demonstrated in Section 5.

**Lemma 4** *Algorithm 2 optimally solves Problem (5).*

**Proof.** The result follows from repeated application of Proposition 3 over the set of nodes $\mathcal{V}$ that has not yet been optimally solved for. ∎

## 4 Complexity of the partitioning algorithm

Linear program (5) can be solved in $O(n^3)$ using interior point methods. Given that the algorithm performs at most $n$ iterations, the worst case complexity of Algorithm 1 is $O(n^4)$. However, the practical complexity of IRP is significantly better than the worst case. Each iteration of LP (5) solves smaller problems. Consider the case of balanced partitioning at each iteration until there are $n$ final blocks. In this case, we can represent the partitioning path as a binary tree with $\log n$ levels, and at each level $k$, LP (5) is solved $2^k$ times on instances of size $\frac{n}{2^k}$ which leads to a total complexity of

$$\sum_{k=0}^{\log n} 2^k \left(\frac{n}{2^k}\right)^3 = n^3 \left(\sum_{k=0}^{\log n} \left(\frac{1}{4}\right)^k\right) = n^3 \left(\frac{1 - .25^{\log n + 1}}{.75}\right),$$

subject to additional constants. For $n \ge 10$, the summation is approximately 1.33, and hence in this case the partitioning algorithm has complexity $O(1.33n^3)$ (considering the complexity of interior

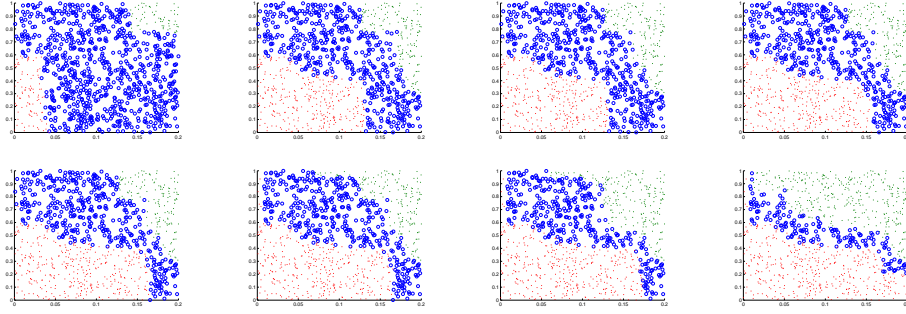

**Figure 1:** Illustration of LP (5) decomposition. Data here is 2 dimensional with only 1000 nodes in order to leave a clear picture. First 7 iterations and the final iteration 16 of the decomposition are shown from left to right and top to bottom. The remaining nodes (blue circles) to identify as $\pm1$ decreases through the iterations. LP (5) solved on the entire set of nodes in the first picture may be too large for memory. Hence subproblems are solved on the lower left (red dots) and upper right (green dots) of the networks and some nodes are fixed from the solution of these subproblems. This is repeated until the number of unidentified nodes in the last iteration is of small enough size for memory. Note that at each iteration the three groups obey isotonicity.

point methods for partitioning). More generally, let $p$ and $1 - p$ be the percentages on each split. Table 1 displays the constants $c$ representing the complexity from $O(cn^3)$ over varying $p$ and $n$. As demonstrated, the problem size rapidly decreases and the complexity is in practice $O(n^3)$.

|          | n=100      | n=1000     | n=10000    |
|----------|------------|------------|------------|
| p=0.55   | $1.35n^3$  | $1.35n^3$  | $1.35n^3$  |
| p=0.65   | $1.46n^3$  | $1.46n^3$  | $1.47n^3$  |
| p=0.75   | $1.77n^3$  | $1.78n^3$  | $1.78n^3$  |
| p=0.85   | $2.56n^3$  | $2.61n^3$  | $2.61n^3$  |
| p=0.95   | $6.41n^3$  | $6.94n^3$  | $7.01n^3$  |

**Table 1:** Complexity: Groups are split with ratio $p$ at each iteration. Complexity in practice is $O(n^3)$.

## 5 Numerical experiments

We here demonstrate that exact isotonic regression is computationally tractable for very large problems, and compare against the time it takes to get an approximation. We first show the computational performance of isotonic regression on simulated data sets as large as $2 \times 10^5$ training points with more than $10^7$ constraints. We then show the favorable predictive performance of isotonic regression on large simulated data sets.

### 5.1 Large-Scale Computations

Figure 2 demonstrates that the partitioning algorithm with decompositions of the partitioning step can solve very large isotonic regressions. Three dimensional data is simulated from $\mathcal{U}(0, 2)$ and the responses are created as linear functions plus noise. The size of the training sets varies from $10^4$ to $2 \times 10^5$ points. The left figure shows that the partitioning algorithm finds the globally optimal isotonic regression solution in not much more time than it takes to find an approximation as done in [6] for very large problems. Although the worst-case complexity of our exact algorithm is much worse, the two algorithms scale comparably in practice.

Figure 2 (right) shows how the number of partitions (left axis) increases as the number of training points increases. It is not clear why the approximation in [6] has less partitions as the size of the problem grows. More partitions (left axis) require solving more network flow problems, however, as discussed, they reduce in size very quickly over the partitioning path, resulting in the practical complexity seen in the figure on the left. The bold black line also shows the number of constraints (right axis) which goes up to more than $10^7$ constraints.

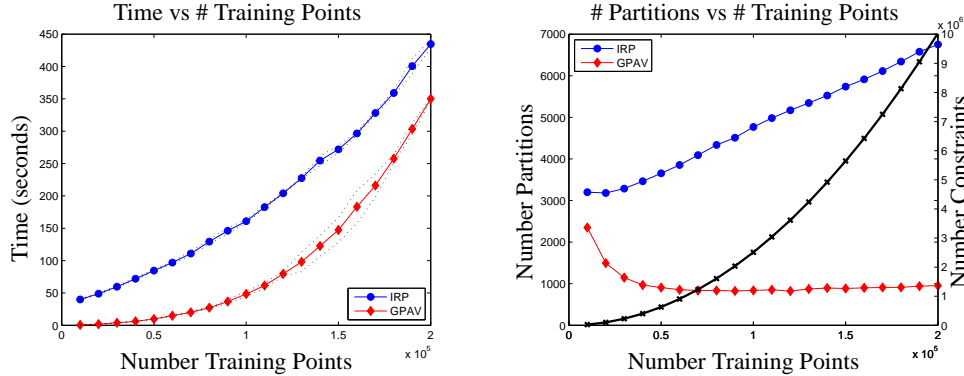

**Figure 2:** IRP performance on large-scale simulations. Data $x \in \mathbf{R}^3$ has $x_i \sim \mathcal{U}(0, 2)$. Responses $y$ are linear functions plus noise. Number of training points varies from $10^4$ to $2 \times 10^5$. Results shown are averages of 5 simulations with dotted lines at $\pm$ one standard deviation. Time (seconds) versus number of training points is on the left. On the right, the number of partitions is illustrated using the left axis and the bold black line shows the average number of constraints per test using the right axis.

## 5.2 Predictive Performance

Here we show that isotonic regression is a useful tool when the data fits the monotonic framework. Data is simulated as above and responses are constructed as $y_i = \prod_i x_i + \mathcal{N}(0, .5^2)$ where $p$ varies from 2 to 6. The training set varies from 500 to 5000 to 50000 points and the test size is fixed at 5000. Results are averaged over 10 trials and 95% confidence intervals are given. A comparison is made between isotonic regression and linear least squares regression. With only 500 training points, the model is poorly fitted and a simple linear regression performs much better. 5000 training points is sufficient to fit the model well with up to 4 dimensions, after which linear regression outperforms the isotonic regression, and 50000 training points fits the model well up with up to 5 dimensions. Two trends are observed. Larger training sets allow better models to be fit which improves performance while higher dimensions increase overfitting which, in turn, decreases performance.

| Dim | IRP MSE n=500 | LS MSE n=500 | IRP MSE n=5000 | LS MSE n=5000 | IRP MSE n=50000 | LS MSE n=50000 |
|---|---|---|---|---|---|---|
| 2 | $0.69 \pm 0.01$ | $\mathbf{0.37 \pm 0.00}$ | $\mathbf{0.27 \pm 0.00}$ | $0.36 \pm 0.00$ | $\mathbf{0.25 \pm 0.00}$ | $0.36 \pm 0.00$ |
| 3 | $0.76 \pm 0.03$ | $\mathbf{0.65 \pm 0.01}$ | $\mathbf{0.31 \pm 0.00}$ | $0.61 \pm 0.01$ | $\mathbf{0.26 \pm 0.00}$ | $0.62 \pm 0.00$ |
| 4 | $1.45 \pm 0.08$ | $\mathbf{1.08 \pm 0.01}$ | $\mathbf{0.61 \pm 0.02}$ | $1.08 \pm 0.02$ | $\mathbf{0.34 \pm 0.01}$ | $1.06 \pm 0.03$ |
| 5 | $4.61 \pm 0.65$ | $\mathbf{1.76 \pm 0.02}$ | $2.61 \pm 0.16$ | $\mathbf{1.88 \pm 0.04}$ | $\mathbf{0.93 \pm 0.04}$ | $1.86 \pm 0.05$ |
| 6 | $12.89 \pm 1.30$ | $\mathbf{3.06 \pm 0.04}$ | $8.41 \pm 1.36$ | $\mathbf{2.84 \pm 0.07}$ | $3.37 \pm 0.06$ | $\mathbf{2.83 \pm 0.12}$ |

**Table 2:** Statistics for simulation generated with $y_i = \prod_i x_i + \mathcal{N}(0, .5^2)$. A comparison between the results of IRP and a least squares linear regression is shown. Bold demonstrates statistical significance at 95% confidence.

## 6  Conclusion

This paper demonstrates that isotonic regression can be used to solve extremely large problems. Fast approximations are useful, however, as shown, globally optimal solutions are also computationally tractable. Indeed, isotonic regression as done here performs with a complexity of $O(n^3)$ in practice. As also shown, isotonic regression performs well at reasonable dimensions, but suffers from over-fitting as the dimension of the data increases. Extensions of this algorithm will analyze the path of partitions in order to control overfitting by stopping the algorithm early. Statistical complexity of the models generated by partitioning will be examined. Furthermore, similar results will be made for isotonic regression with different loss functions.

# References

[1] R.E. Barlow and H.D. Brunk. The isotonic regression problem and its dual. *Journal of the American Statistical Association*, 67(337):140–147, 1972.

[2] G. Obozinski, G. Lanckriet, C. Grant, M.I. Jordan, and W.S. Noble. Consistent probabilistic outputs for protein function prediction. *Genome Biology*, 9:247–254, 2008. Open Access.

[3] M.J. Schell and B. Singh. The reduced monotonic regression method. *Journal of the American Statistical Association*, 92(437):128–135, 1997.

[4] J.B. Kruskal. Multidimensional scaling by optimizing goodness of fit to a nonmetric hypothesis. *Psychometrika*, 29(1), 1964.

[5] H. Block, S. Qian, and A. Sampson. Structure algorithms for partially ordered isotonic regression. *Journal of Computational and Graphical Statistcs*, 3(3):285–300, 1994.

[6] O. Burdakov, O. Sysoev, A. Grimvall, and M. Hussian. An $o(n^2)$ algorithm for isotonic regression. 83:25–83, 2006. In: G. Di Pillo and M. Roma (Eds) *Large-Scale Nonlinear Optimization*. Series: Nonconvex Optimization and Its Applications.

[7] C.-I. C. Lee. The min-max algorithm and isotonic regression. *The Annals of Statistics*, 11(2):467–477, 1983.

[8] J. de Leeuw, K. Hornik, and P. Mair. Isotone optimization in r: Pool-adjacent-violators algorithm (pava) and active set methods. 2009. UC Los Angeles: Department of Statistics, UCLA. Retrieved from: http://cran.r-project.org/web/packages/isotone/vignettes/isotone.pdf.

[9] W.L. Maxwell and J.A. Muckstadt. Establishing consistent and realistic reorder intervals in production-distribution systems. *Operations Research*, 33(6):1316–1341, 1985.

[10] R.D.C. Monteiro and I. Adler. Interior path following primal-dual algorithms. part II: Convex quadratic programming. *Mathematical Programming*, 44:43–66, 1989.

[11] O. Burdakov, O. Sysoev, and A. Grimvall. Generalized PAV algorithm with block refinement for partially ordered monotonic regression. pages 23–37, 2009. In: A. Feelders and R. Potharst (Eds.) Proc. of the Workshop on Learning Monotone Models from Data at the European Conference on Machine Learning and Principles and Practice of Knowledge Discovery in Databases.

[12] P.M. Pardalos and G. Xue. Algorithms for a class of isotonic regression problems. *Algorithmica*, 23:211–222, 1999.

[13] K.G. Murty. *Linear Programming*. John Wiley & Sons, Inc., 1983.

[14] R. Chandrasekaran, Y.U. Ryu, V.S. Jacob, and S. Hong. Isotonic separation. *INFORMS Journal on Computing*, 17(4):462–474, 2005.

[15] MOSEK ApS. The MOSEK optimization tools manual. version 6.0, revision 61. 2010. Software available at http://www.mosek.com.

[16] R. K. Ahuja, T. L. Magnanti, and J. B. Orlin. *Network Flows: Theory, Algorithms, and Applications*. Prentice-Hall, Inc., 1993.

